# A Probabilistic Interpretation of SVMs with an Application to Unbalanced Classification

**Yves Grandvalet** [*]
Heudiasyc, CNRS/UTC
60205 Compiègne cedex, France
grandval@utc.fr

**Johnny Mariéthoz   Samy Bengio**
IDIAP Research Institute
1920 Martigny, Switzerland
{marietho,bengio}@idiap.ch

## Abstract

In this paper, we show that the hinge loss can be interpreted as the neg-log-likelihood of a semi-parametric model of posterior probabilities. From this point of view, SVMs represent the parametric component of a semi-parametric model fitted by a maximum a posteriori estimation procedure. This connection enables to derive a mapping from SVM scores to estimated posterior probabilities. Unlike previous proposals, the suggested mapping is interval-valued, providing a set of posterior probabilities compatible with each SVM score. This framework offers a new way to adapt the SVM optimization problem to unbalanced classification, when decisions result in unequal (asymmetric) losses. Experiments show improvements over state-of-the-art procedures.

## 1  Introduction

In this paper, we show that support vector machines (SVMs) are the solution of a relaxed maximum a posteriori (MAP) estimation problem. This relaxed problem results from fitting a semi-parametric model of posterior probabilities. This model is decomposed into two components: the parametric component, which is a function of the SVM score, and the non-parametric component which we call a nuisance function. Given a proper binding of the nuisance function adapted to the considered problem, this decomposition enables to concentrate on selected ranges of the probability spectrum. The estimation process can thus allocate model capacity to the neighborhoods of decision boundaries.

The connection to semi-parametric models provides a probabilistic interpretation of SVM scores, which may have several applications, such as estimating confidences over the predictions, or dealing with unbalanced losses. (which occur in domains such as diagnosis, intruder detection, etc). Several mappings relating SVM scores to probabilities have already been proposed (Sollich 2000, Platt 2000), but they are subject to arbitrary choices, which are avoided here by their integration to the nuisance function.

The paper is organized as follows. Section 2 presents the semi-parametric modeling approach; Section 3 shows how we reformulate SVM in this framework; Section 4 proposes several outcomes of this formulation, including a new method to handle unbalanced losses, which is tested empirically in Section 5. Finally, Section 6 briefly concludes the paper.

---

[*]This work was supported in part by the IST Programme of the European Community, under the PASCAL Network of Excellence IST-2002-506778. This publication only reflects the authors' views.

## 2 Semi-Parametric Classification

We address the binary classification problem of estimating a decision rule from a learning set $\mathcal{L}_n = \{(\mathbf{x}_i, y_i)\}_{i=1}^n$, where the $i$th example is described by the pattern $\mathbf{x}_i \in \mathcal{X}$ and the associated response $y_i \in \{-1, 1\}$. In the framework of maximum likelihood estimation, classification can be addressed either via generative models, *i.e.* models of the joint distribution $P(X, Y)$, or via discriminative methods modeling the conditional $P(Y|X)$.

### 2.1 Complete and Marginal Likelihood, Nuisance Functions

Let $p(1|\mathbf{x}; \boldsymbol{\theta})$ denote the model of $P(Y = 1|X = \mathbf{x})$, $p(\mathbf{x}; \boldsymbol{\psi})$ the model of $P(X)$ and $t_i$ the binary response variable such that $t_i = 1$ when $y_i = 1$ and $t_i = 0$ when $y_i = -1$. Assuming independent examples, the complete log-likelihood can be decomposed as

$$L(\boldsymbol{\theta}, \boldsymbol{\psi}; \mathcal{L}_n) = \sum_i t_i \log(p(1|\mathbf{x}_i; \boldsymbol{\theta})) + (1 - t_i) \log(1 - p(1|\mathbf{x}_i; \boldsymbol{\theta})) + \log(p(\mathbf{x}_i; \boldsymbol{\psi})) \ , \ (1)$$

where the two first terms of the right-hand side represent the marginal or conditional likelihood, that is, the likelihood of $p(1|\mathbf{x}; \boldsymbol{\theta})$.

For classification purposes, the parameter $\boldsymbol{\psi}$ is not relevant, and may thus be qualified as a nuisance parameter (Lindsay 1985). When $\boldsymbol{\theta}$ can be estimated independently of $\boldsymbol{\psi}$, maximizing the marginal likelihood provides the estimate returned by maximizing the complete likelihood with respect to $\boldsymbol{\theta}$ and $\boldsymbol{\psi}$. In particular, when no assumption whatsoever is made on $P(X)$, maximizing the conditional likelihood amounts to maximize the joint likelihood (McLachlan 1992). The density of inputs is then considered as a nuisance function.

### 2.2 Semi-Parametric Models

Again, for classification purposes, estimating $P(Y|X)$ may be considered as too demanding. Indeed, taking a decision only requires the knowledge of $\text{sign}(2P(Y = 1|X = \mathbf{x}) - 1)$. We may thus consider looking for the decision rule minimizing the empirical classification error, but this problem is intractable for non-trivial models of discriminant functions.

Here, we briefly explore how semi-parametric models (Oakes 1988) may be used to reduce the modelization effort as compared to the standard likelihood approach. For this, we consider a two-component semi-parametric model of $P(Y = 1|X = \mathbf{x})$, defined as $p(1|\mathbf{x}; \boldsymbol{\theta}) = g(\mathbf{x}; \boldsymbol{\theta}) + \varepsilon(\mathbf{x})$, where the parametric component $g(\mathbf{x}; \boldsymbol{\theta})$ is the function of interest, and where the non-parametric component $\varepsilon$ is a constrained nuisance function. Then, we address the maximum likelihood estimation of the semi-parametric model $p(1|\mathbf{x}; \boldsymbol{\theta})$

$$\begin{cases} \min_{\boldsymbol{\theta}, \varepsilon} & -\sum_i t_i \log(p(1|\mathbf{x}_i; \boldsymbol{\theta})) + (1 - t_i) \log(1 - p(1|\mathbf{x}_i; \boldsymbol{\theta})) \\ \text{s. t.} & p(1|\mathbf{x}; \boldsymbol{\theta}) = g(\mathbf{x}; \boldsymbol{\theta}) + \varepsilon(\mathbf{x}) \\ & 0 \le p(1|\mathbf{x}; \boldsymbol{\theta}) \le 1 \\ & \varepsilon^-(\mathbf{x}) \le \varepsilon(\mathbf{x}) \le \varepsilon^+(\mathbf{x}) \end{cases} \quad (2)$$

where $\varepsilon^-$ and $\varepsilon^+$ are user-defined functions, which place constraints on the non-parametric component $\varepsilon$. According to these constraints, one pursues different objectives, which can be interpreted as either weakened or focused versions of the original problem of estimating precisely $P(Y|X)$ on the whole range $[0, 1]$.

At the one extreme, when $\varepsilon^- = \varepsilon^+$, one recovers a parametric maximum likelihood problem, where the estimate of posterior probabilities $p(1|\mathbf{x}; \boldsymbol{\theta})$ is simply $g(\mathbf{x}; \boldsymbol{\theta})$ shifted by the baseline function $\varepsilon$. At the other extreme, when $\varepsilon^-(\mathbf{x}) \le -g(\mathbf{x})$ and $\varepsilon^+(\mathbf{x}) \ge 1 - g(\mathbf{x})$, $p(1|\cdot; \boldsymbol{\theta})$ perfectly explains (interpolates) any training sample for any $\boldsymbol{\theta}$, and the optimization problem in $\boldsymbol{\theta}$ is ill-posed. Note that the optimization problem in $\varepsilon$ is always ill-posed, but this is not of concern as we do not wish to estimate the nuisance function.

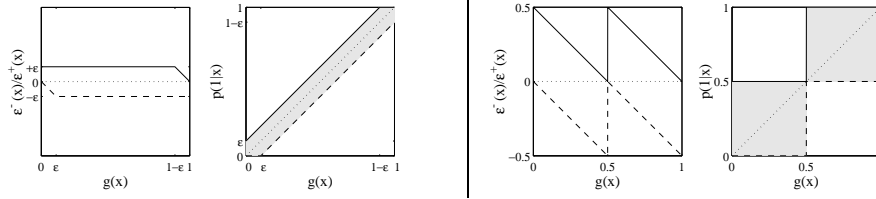

Figure 1: Two examples of $\varepsilon^-(\mathbf{x})$ (dashed) and $\varepsilon^+(\mathbf{x})$ (plain) *vs.* $g(\mathbf{x})$ and resulting $\epsilon$-tube of possible values for the estimate of $P(Y = 1 | X = \mathbf{x})$ (gray zone) *vs.* $g(\mathbf{x})$.

Generally, as $\varepsilon$ is not estimated, the estimate of posterior probabilities $p(1|\mathbf{x}; \boldsymbol{\theta})$ is only known to lie within the interval $[g(\mathbf{x}; \boldsymbol{\theta}) + \varepsilon^-(\mathbf{x}), g(\mathbf{x}; \boldsymbol{\theta}) + \varepsilon^+(\mathbf{x})]$. In what follows, we only consider functions $\varepsilon^-$ and $\varepsilon^+$ expressed as functions of the argument $g(\mathbf{x})$, for which the interval can be recovered from $g(\mathbf{x})$ alone. We also require $\varepsilon^-(\mathbf{x}) \leq 0 \leq \varepsilon^+(\mathbf{x})$, in order to ensure that $g(\mathbf{x}; \boldsymbol{\theta})$ is an admissible value of $p(1|\mathbf{x}; \boldsymbol{\theta})$.

Two simple examples are displayed in Figure 1. The two first graphs represent $\varepsilon^-$ and $\varepsilon^+$ designed to estimate posterior probabilities up to precision $\epsilon$, and the corresponding $\epsilon$-tube of admissible estimates knowing $g(\mathbf{x})$. The two last graphs represent the same functions for $\varepsilon^-$ and $\varepsilon^+$ defined to focus on the only relevant piece of information regarding decision: estimating where $P(Y|X)$ is above $1/2$. [1]

## 2.3 Estimation of the Parametric Component

The definitions of $\varepsilon^-$ and $\varepsilon^+$ affect the estimation of the parametric component. Regarding $\boldsymbol{\theta}$, when the values of $g(\mathbf{x}; \boldsymbol{\theta}) + \varepsilon^-(\mathbf{x})$ and $g(\mathbf{x}; \boldsymbol{\theta}) + \varepsilon^+(\mathbf{x})$ lie within $[0, 1]$, problem (2) is equivalent to the following relaxed maximum likelihood problem

$$\begin{cases} \min\limits_{\boldsymbol{\theta}, \boldsymbol{\varepsilon}} & -\sum_i t_i \log(g(\mathbf{x}_i; \boldsymbol{\theta}) + \varepsilon_i) + (1 - t_i) \log(1 - g(\mathbf{x}_i; \boldsymbol{\theta}) - \varepsilon_i) \\ \text{s. t.} & \varepsilon^-(\mathbf{x}_i) \leq \varepsilon_i \leq \varepsilon^+(\mathbf{x}_i) \quad i = 1, \dots, n \end{cases} \tag{3}$$

where $\boldsymbol{\varepsilon}$ is an $n$-dimensional vector of slack variables. The problem is qualified as relaxed compared to the the maximum likelihood estimation of posterior probabilities by $g(\mathbf{x}_i; \boldsymbol{\theta})$, because modeling posterior probabilities by $g(\mathbf{x}_i; \boldsymbol{\theta}) + \varepsilon_i$ is a looser objective.

The monotonicity of the objective function with respect to $\varepsilon_i$ implies that the constraints $\varepsilon^-(\mathbf{x}_i) \leq \varepsilon_i$ and $\varepsilon_i \leq \varepsilon^+(\mathbf{x}_i)$ are saturated at the solution of (3) for $t_i = 0$ or $t_i = 1$ respectively. Thus, the loss in (3) is the neg-log-likelihood of the lower or the upper bound on $p(1|\mathbf{x}_i; \boldsymbol{\theta})$ respectively. Provided that $g$, $\varepsilon^-$ and $\varepsilon^+$ are defined such that $\varepsilon^-(\mathbf{x}) \leq \varepsilon^+(\mathbf{x})$, $0 \leq g(\mathbf{x}) + \varepsilon^-(\mathbf{x}) \leq 1$ and $0 \leq g(\mathbf{x}) + \varepsilon^+(\mathbf{x}) \leq 1$, the optimization problem with respect to $\boldsymbol{\theta}$ reduces to

$$\min_{\boldsymbol{\theta}} -\sum_i t_i \log(g(\mathbf{x}_i; \boldsymbol{\theta}) + \varepsilon^+(\mathbf{x}_i)) + (1 - t_i) \log(1 - g(\mathbf{x}_i; \boldsymbol{\theta}) - \varepsilon^-(\mathbf{x}_i)) \ . \tag{4}$$

Figure 2 displays the losses for positive examples corresponding to the choices of $\varepsilon^-$ and $\varepsilon^+$ depicted in Figure 1 (the losses are symmetrical around 0.5 for negative examples). Note that the convexity of the objective function with respect to $g$ depends on the choices of $\varepsilon^-$ and $\varepsilon^+$. One can show that, providing $\varepsilon^+$ and $\varepsilon^-$ are respectively concave and convex functions of $g$, then the loss (4) is convex in $g$.

When $\varepsilon^-(\mathbf{x}) \leq 0 \leq \varepsilon^+(\mathbf{x})$, $g(\mathbf{x})$ is an admissible estimate of $P(Y = 1|\mathbf{x})$. However, the relaxed loss (4) is optimistic, below the neg-log-likelihood of $g$. This optimism usually

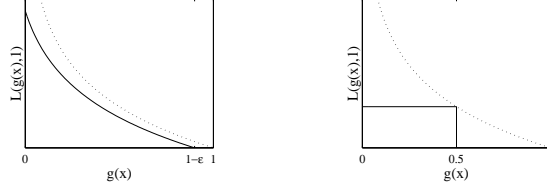

Figure 2: Losses for positive examples (plain) and neg-log-likelihood of $g(\mathbf{x})$ (dotted) *vs.* $g(\mathbf{x})$. Left: for the function $\varepsilon^+$ displayed on the left-hand side of Figure 1; right: for the function $\varepsilon^+$ displayed on the right-hand side of Figure 1.

results in a non-consistent estimation of posterior probabilities (*i.e* $g(\mathbf{x})$ does not converge towards $P(Y = 1|X = \mathbf{x})$ as the sample size goes to infinity), a common situation in semi-parametric modeling (Lindsay 1985). This lack of consistency should not be a concern here, since the non-parametric component is purposely introduced to address a looser estimation problem. We should therefore restrict consistency requirements to the primary goal of having posterior probabilities in the $\epsilon$-tube $[g(\mathbf{x}) + \varepsilon^-(\mathbf{x}), g(\mathbf{x}) + \varepsilon^+(\mathbf{x})]$.

## 3  Semi-Parametric Formulation of SVMs

Several authors pointed the closeness of SVM and the MAP approach to Gaussian processes (Sollich (2000) and references therein). However, this similarity does not provide a proper mapping from SVM scores to posterior probabilities. Here, we resolve this difficulty thanks to the additional degrees of freedom provided by semi-parametric modelling.

### 3.1  SVMs and Gaussian Processes

In its primal Lagrangian formulation, the SVM optimization problem reads

$$\min_{f,b} \frac{1}{2}\|f\|_{\mathcal{H}}^2 + C \sum_i [1 - y_i(f(\mathbf{x}_i) + b)]_+ \ , \tag{5}$$

where $\mathcal{H}$ is a reproducing kernel Hilbert space with norm $\| \cdot \|_{\mathcal{H}}$, $C$ is a regularization parameter and $[f]_+ = \max(f, 0)$.

The penalization term in (5) can be interpreted as a Gaussian prior on $f$, with a covariance function proportional to the reproducing kernel of $\mathcal{H}$ (Sollich 2000). Then, the interpretation of the hinge loss as a marginal log-likelihood requires to identify an affine function of the last term of (5) with the two first terms of (1). We thus look for two constants $c_0$ and $c_1 \neq 0$, such that, for all values of $f(\mathbf{x}) + b$, there exists a value $0 \leq p(1|\mathbf{x}) \leq 1$ such that

$$\begin{cases} p(1|\mathbf{x}) & = & \exp -(c_0 + c_1[1 - (f(\mathbf{x}) + b)]_+) \\ 1 - p(1|\mathbf{x}) & = & \exp -(c_0 + c_1[1 + (f(\mathbf{x}) + b)]_+) \end{cases} \ . \tag{6}$$

The system (6) has a solution over the whole range of possible values of $f(\mathbf{x}) + b$ if and only if $c_0 = \log(2)$ and $c_1 = 0$. Thus, the SVM optimization problem does not implement the MAP approach to Gaussian processes.

To proceed with a probabilistic interpretation of SVMs, Sollich (2000) proposed a normalized probability model. The normalization functional was chosen arbitrarily, and the consequences of this choice on the probabilistic interpretation was not evaluated. In what follows, we derive an imprecise mapping, with interval-valued estimates of probabilities, representing the set of all admissible semi-parametric formulations of SVM scores.

### 3.2  SVMs and Semi-Parametric Models

With the semi-parametric models of Section 2.2, one has to identify an affine function of the hinge loss with the two terms of (4). Compared to the previous situation, one has the

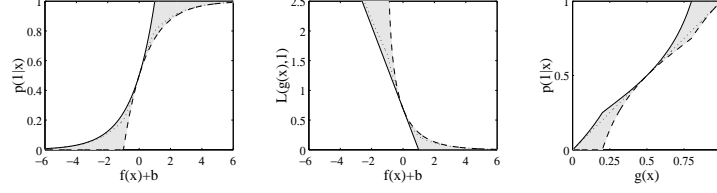

Figure 3: Left: lower (dashed) and upper (plain) posterior probabilities $[g(\mathbf{x}) + \varepsilon^-(\mathbf{x}), g(\mathbf{x}) + \varepsilon^+(\mathbf{x})]$ *vs.* SVM scores $f(\mathbf{x}) + b$; center: corresponding neg-log-likelihood of $g(\mathbf{x})$ for positive examples *vs.* $f(\mathbf{x})+b$. right: lower (dashed) and upper (plain) posterior probabilities *vs.* $g(\mathbf{x})$, for $g$ defined in (8).

freedom to define the slack functions $\varepsilon^-$ and $\varepsilon^+$. The identification problem is now

$$
\begin{cases}
g(\mathbf{x}) + \varepsilon^+(\mathbf{x}) = \exp-(c_0 + c_1[1 - (f(\mathbf{x}) + b)]_+) \\
1 - g(\mathbf{x}) - \varepsilon^-(\mathbf{x}) = \exp-(c_0 + c_1[1 + (f(\mathbf{x}) + b)]_+) \\
\text{s.t.} \quad 0 \le g(\mathbf{x}) + \varepsilon^-(\mathbf{x}) \le 1 \\
\qquad\; 0 \le g(\mathbf{x}) + \varepsilon^+(\mathbf{x}) \le 1 \\
\qquad\; \varepsilon^-(\mathbf{x}) \le \varepsilon^+(\mathbf{x})
\end{cases} \tag{7}
$$

Provided $c_0 = 0$ and $0 < c_1 \le \log(2)$, there are functions $g$, $\varepsilon^-$ and $\varepsilon^+$ such that the above problem has a solution. Hence, we obtain a set of probabilistic interpretations fully compatible with SVM scores. The solutions indexed by $c_1$ are nested, in the sense that, for any $\mathbf{x}$, the length of the uncertainty interval, $\varepsilon^+(\mathbf{x}) - \varepsilon^-(\mathbf{x})$, is monotonically decreasing in $c_1$: the interpretation of SVM scores as posterior probabilities gets tighter as $c_1$ increases.

The most restricted subset of admissible interpretations, with the shortest uncertainty intervals, obtained for $c_1 = \log(2)$, is represented in the left-hand side of Figure 3. The loss incurred by a positive example is represented on the central graph, where the gray zone represents the neg-log-likelihood of all admissible solutions of $g(\mathbf{x})$. Note that the hinge loss is proportional to the neg-log-likelihood of the upper posterior probability $g(\mathbf{x}) + \varepsilon^+(\mathbf{x})$, which is the loss for positive examples in the semi-parametric model in (4). Conversely, the hinge loss for negative examples is reached for $g(\mathbf{x}) + \varepsilon^-(\mathbf{x})$. An important observation, that will be useful in Section 4.2 is that the neg-log-likelihood of any admissible functions $g(\mathbf{x})$ is tangent to the hinge loss at $f(\mathbf{x}) + b = 0$.

The solution is unique in terms of the admissible interval $[g + \varepsilon^-, g + \varepsilon^+]$, but many definitions of $(\varepsilon^-, \varepsilon^+, g)$ solve (7). For example, $g$ may be defined as

$$
g(\mathbf{x}; \boldsymbol{\theta}) = \frac{2^{-[1-(f(\mathbf{x})+b)]_+}}{2^{-[1+(f(\mathbf{x})+b)]_+} + 2^{-[1-(f(\mathbf{x})+b)]_+}} , \tag{8}
$$

which is essentially the posterior probability model proposed by Sollich (2000), represented dotted in the first two graphs of Figure 3.

The last graph of Figure 3 displays the mapping from $g(\mathbf{x})$ to admissible values of $p(1|\mathbf{x})$ which results from the choice described in (8). Although the interpretation of SVM scores does not require to specify $g$, it may worth to list some features common to all options. First, $g(\mathbf{x}) + \varepsilon^-(\mathbf{x}) = 0$ for all $g(\mathbf{x})$ below some threshold $g_0 > 0$, and conversely, $g(\mathbf{x}) + \varepsilon^+(\mathbf{x}) = 1$ for all $g(\mathbf{x})$ above some threshold $g_1 < 1$. These two features are responsible for the sparsity of the SVM solution. Second, the estimation of posterior probabilities is accurate at $0.5$, and the length of the uncertainty interval on $p(1|\mathbf{x})$ monotonically increases in $[g_0, 0.5]$ and then monotonically decreases in $[0.5, g_1]$. Hence, the training objective of SVMs is intermediate between the accurate estimation of posterior probabilities on the whole range $[0, 1]$ and the minimization of the classification risk.

## 4   Outcomes of the Probabilistic Interpretation

This section gives two consequences of our probabilistic interpretation of SVMs. Further outcomes, still reserved for future research are listed in Section 6.

### 4.1   Pointwise Posterior Probabilities from SVM Scores

Platt (2000) proposed to estimate posterior probabilities from SVM scores by fitting a logistic function over the SVM scores. The only logistic function compatible with the most stringent interpretation of SVMs in the semi-parametric framework,

$$g(\mathbf{x}; \boldsymbol{\theta}) = \frac{1}{1 + 4^{-(f(\mathbf{x})+b))}} \ , \tag{9}$$

is identical to the model of Sollich (2000) (8) when $f(\mathbf{x}) + b$ lies in the interval $[-1, 1]$.

Other logistic functions are compatible with the looser interpretations obtained by letting $c_1 < \log(2)$, but their use as pointwise estimates is questionable, since the associated confidence interval is wider. In particular, the looser interpretations do not ensure that $f(\mathbf{x}) + b = 0$ corresponds to $g(\mathbf{x}) = 0.5$. Then, the decision function based on the estimated posterior probabilities by $g(\mathbf{x})$ may differ from the SVM decision function.

Being based on an arbitrary choice of $g(\mathbf{x})$, pointwise estimates of posterior probabilities derived from SVM scores should be handled with caution. As discussed by Zhang (2004), they may only be consistent at $f(\mathbf{x}) + b = 0$, where they may converge towards $0.5$.

### 4.2   Unbalanced Classification Losses

SVMs are known to perform well regarding misclassification error, but they provide skewed decision boundaries for unbalanced classification losses, where the losses associated with incorrect decisions differ according to the true label. The mainstream approach used to address this problem consists in using different losses for positive and negative examples (Morik et al. 1999, Veropoulos et al. 1999), *i.e.*

$$\min_{f,b} \frac{1}{2}\|f\|_{\mathcal{H}}^2 + C^+ \sum_{\{i|y_i=1\}} [1 - (f(\mathbf{x}_i)+b)]_+ + C^- \sum_{\{i|y_i=-1\}} [1 + (f(\mathbf{x}_i)+b)]_+ \ , \tag{10}$$

where the coefficients $C^+$ and $C^-$ are constants, whose ratio is equal to the ratio of the losses $\ell_{\mathrm{FN}}$ and $\ell_{\mathrm{FP}}$ pertaining to false negatives and false positives, respectively (Lin et al. 2002).[2] Bayes' decision theory defines the optimal decision rule by positive classification when $P(y = 1|\mathbf{x}) > P_0$, where $P_0 = \frac{\ell_{\mathrm{FP}}}{\ell_{\mathrm{FP}}+\ell_{\mathrm{FN}}}$. We may thus rewrite $C^+ = C \cdot (1 - P_0)$ and $C^- = C \cdot P_0$. With such definitions, the optimization problem may be interpreted as an upper-bound on the classification risk defined from $\ell_{\mathrm{FN}}$ and $\ell_{\mathrm{FP}}$. However, the machinery of Section 3.2 unveils a major problem: the SVM decision function provided by $\mathrm{sign}(f(\mathbf{x}_i) + b)$ is not consistent with the probabilistic interpretation of SVM scores.

We address this problem by deriving another criterion, by requiring that the neg-log-likelihood of any admissible functions $g(\mathbf{x})$ is tangent to the hinge loss at $f(\mathbf{x}) + b = 0$. This leads to the following problem:

$$\min_{f,b} \frac{1}{2}\|f\|_{\mathcal{H}}^2 \quad + \quad C \left( \sum_{\{i|y_i=1\}} [-\log(P_0) - (1 - P_0)(f(\mathbf{x}_i) + b)]_+ + \right.$$

$$\left. \sum_{\{i|y_i=-1\}} [-\log(1 - P_0) + P_0(f(\mathbf{x}_i) + b)]_+ \right) \ . \tag{11}$$

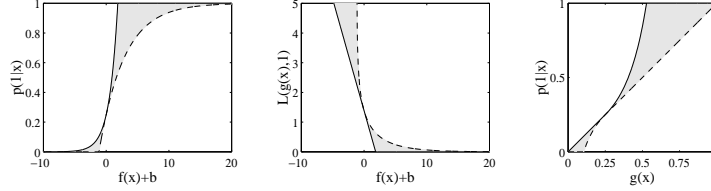

Figure 4: Left: lower (dashed) and upper (plain) posterior probabilities $[g(\mathbf{x}) + \varepsilon^-(\mathbf{x}), g(\mathbf{x}) + \varepsilon^+(\mathbf{x})]$ *vs.* SVM scores $f(\mathbf{x}) + b$ obtained from (11) with $P_0 = 0.25$; center: corresponding neg-log-likelihood of $g(\mathbf{x})$ for positive examples *vs.* $f(\mathbf{x}) + b$. right: lower (dashed) and upper (plain) posterior probabilities *vs.* $g(\mathbf{x})$, for $g$ defined by $\varepsilon^+(\mathbf{x}) = 0$ for $f(\mathbf{x}) + b \leq 0$ and $\varepsilon^-(\mathbf{x}) = 0$ for $f(\mathbf{x}) + b \geq 0$.

This loss differs from (10), in the respect that the margin for positive examples is smaller than the one for negative examples when $P_0 < 0.5$. In particular, (10) does not affect the SVM solution for separable problems, while in (11), the decision boundary moves towards positive support vectors when $P_0$ decreases. The analogue of Figure 3, displayed on Figure 4, shows that one recovers the characteristics of the standard SVM loss, except that the focus is now on the posterior probability $P_0$ defined by Bayes' decision rule.

## 5  Experiments with Unbalanced Classifications Losses

It is straightforward to implement (11) in standard SVM packages. For experimenting with difficult unbalanced two-class problems, we used the Forest database, the largest available UCI dataset (`http://kdd.ics.uci.edu/databases/covertype/`). We consider the subproblem of discriminating the positive class Krummholz (20510 examples) against the negative class Spruce/Fir (211840 examples). The ratio of negative to positive examples is high, a feature commonly encountered with unbalanced classification losses.

The training set was built by random selection of size 11 000 (1000 and 10 000 examples from the positive and negative class respectively); a validation set, of size 11 000 was drawn identically among the other examples; finally, the test set, of size 99 000, was drawn among the remaining examples.

The performance was measured by the weighted risk function $R = \frac{1}{n}(N_{\mathrm{FN}}\ell_{\mathrm{FN}} + N_{\mathrm{FP}}\ell_{\mathrm{FP}})$, where $N_{\mathrm{FN}}$ and $N_{\mathrm{FP}}$ are the number of false negatives and false positives, respectively. The loss $\ell_{\mathrm{FP}}$ was set to one, and $\ell_{\mathrm{FN}}$ was successively set to 1, 10 and 100, in order to penalize more and more heavily errors from the under-represented class.

All approaches were tested using SVMs with a Gaussian kernel on normalized data. The hyper-parameters were tuned on the validation set for each of the $\ell_{\mathrm{FN}}$ values. We additionally considered three tuning for the bias $b$: $\hat{b}$ is the bias returned by the algorithm; $\hat{b}_v$ the bias returned by minimizing $R$ on the validation set, which is an optimistic estimate of the bias that could be computed by cross-validation. We also provide results for $b^*$, the optimal bias computed on the test set. This "crystal ball" tuning may not represent an achievable goal, but it shows how far we are from the optimum. Table 1 compares the risk $R$ obtained with the three approaches for the different values of $\ell_{\mathrm{FN}}$.

The first line, with $\ell_{\mathrm{FN}} = 1$ corresponds to the standard classification error, where all training criteria are equivalent in theory and in practice. The bias returned by the algorithm is very close to the optimal one. For $\ell_{\mathrm{FN}} = 10$ and $\ell_{\mathrm{FN}} = 100$, the models obtained by optimizing $C^+/C^-$ (10) and $P_0$ (11) achieve better results than the baseline with the crystal ball bias. While the solutions returned by $C^+/C^-$ can be significantly improved

Table 1: Errors for 3 different criteria and for 3 different models over the Forest database

| $\ell_{\text{FN}}$ | Baseline, problem (5) | | $C^+/C^-$, problem (10) | | | $P_0$, problem (11) | | |
|---|---|---|---|---|---|---|---|---|
| | $\hat{b}$ | $b^*$ | $\hat{b}$ | $\hat{b}_v$ | $b^*$ | $\hat{b}$ | $\hat{b}_v$ | $b^*$ |
| 1 | 0.027 | 0.026 | 0.027 | 0.027 | 0.026 | 0.027 | 0.027 | 0.026 |
| 10 | 0.167 | 0.108 | 0.105 | 0.104 | 0.094 | 0.095 | 0.104 | 0.094 |
| 100 | 1.664 | 0.406 | 0.403 | 0.291 | 0.289 | 0.295 | 0.291 | 0.289 |

by tuning the bias, our criterion provides results that are very close to the optimum, in the range of the performances obtained with the bias optimized on an independant validation set. The new optimization criterion can thus outperform standard approaches for highly unbalanced problems.

## 6  Conclusion

This paper introduced a semi-parametric model for classification which provides an interesting viewpoint on SVMs. The non-parametric component provides an intuitive means of transforming the likelihood into a decision-oriented criterion. This framework was used here to propose a new parameterization of the hinge loss, dedicated to unbalanced classification problems, yielding significant improvements over the classical procedure.

Among other prospectives, we plan to apply the same framework to investigate hinge-like criteria for decision rules including a reject option, where the classifier abstains when a pattern is ambiguous. We also aim at defining losses encouraging sparsity in probabilistic models, such as kernelized logistic regression. We could thus build sparse probabilistic classifiers, providing an accurate estimation of posterior probabilities on a (limited) predefined range of posterior probabilities. In particular, we could derive decision-oriented criteria for multi-class probabilistic classifiers. For example, minimizing classification error only requires to find the class with highest posterior probability, and this search does not require precise estimates of probabilities outside the interval $[1/K, 1/2]$, where $K$ is the number of classes.

## Footnotes

[1] Of course, this naive attempt to minimize the training classification error is doomed to failure. Reformulating the problem does not affect its convexity: it remains NP-hard.

[2]False negatives/positives respectively designate positive/negative examples incorrectly classified.

## References

Y. Lin, Y. Lee, and G. Wahba. Support vector machines for classification in non-standard situations. *Machine Learning*, 46:191–202, 2002.

B. G. Lindsay. Nuisance parameters. In S. Kotz, C. B. Read, and D. L. Banks, editors, *Encyclopedia of Statistical Sciences*, volume 6. Wiley, 1985.

G. J. McLachlan. *Discriminant analysis and statistical pattern recognition*. Wiley, 1992.

K. Morik, P. Brockhausen, and T. Joachims. Combining statistical learning with a knowledge-based approach - a case study in intensive care monitoring. In *Proceedings of ICML*, 1999.

D. Oakes. Semi-parametric models. In S. Kotz, C. B. Read, and D. L. Banks, editors, *Encyclopedia of Statistical Sciences*, volume 8. Wiley, 1988.

J. C. Platt. Probabilities for SV machines. In A. J. Smola, P. L. Bartlett, B. Schölkopf, and D. Schuurmans, editors, *Advances in Large Margin Classifiers*, pages 61–74. MIT Press, 2000.

P. Sollich. Probabilistic methods for support vector machines. In S. A. Solla, T. K. Leen, and K.-R. Müller, editors, *Advances in Neural Information Processing Systems 12*, pages 349–355, 2000.

K. Veropoulos, C. Campbell, and N. Cristianini. Controlling the sensitivity of support vector machines. In T. Dean, editor, *Proc. of the IJCAI*, pages 55–60, 1999.

T. Zhang. Statistical behavior and consistency of classification methods based on convex risk minimization. *Annals of Statistics*, 32(1):56–85, 2004.
